# Economic Properties of Social Networks

**Sham M. Kakade**     **Michael Kearns**     **Luis E. Ortiz**
**Robin Pemantle**     **Siddharth Suri**

University of Pennsylvania
Philadelphia, PA 19104

## Abstract

We examine the marriage of recent probabilistic generative models for social networks with classical frameworks from mathematical economics. We are particularly interested in how the statistical structure of such networks influences global economic quantities such as price variation. Our findings are a mixture of formal analysis, simulation, and experiments on an international trade data set from the United Nations.

## 1   Introduction

There is a long history of research in economics on mathematical models for exchange markets, and the existence and properties of their equilibria. The work of Arrow and Debreu [1954], who established equilibrium existence in a very general commodities exchange model, was certainly one of the high points of this continuing line of inquiry. The origins of the field go back at least to Fisher [1891].

While there has been relatively recent interest in network models for interaction in economics (see Jackson [2003] for a good review), it was only quite recently that a network or graph-theoretic model that generalizes the classical Arrow-Debreu and Fisher models was introduced (Kakade et al. [2004]). In this model, the edges in a network over individual consumers (for example) represent those pairs of consumers that can engage in direct trade. As such, the model captures the many real-world settings that can give rise to limitations on the trading partners of individuals (regulatory restrictions, social connections, embargoes, and so on). In addition, variations in the price of a good can arise due to the topology of the network: certain individuals may be relatively favored or cursed by their position in the graph.

In a parallel development over the last decade or so, there has been an explosion of interest in what is broadly called *social network theory* — the study of apparently "universal" properties of natural networks (such as small diameter, local clustering of edges, and heavy-tailed distribution of degree), and statistical generative models that explain such properties. When viewed as economic networks, the assumptions of individual rationality in these works are usually either non-existent, or quite weak, compared to the Arrow-Debreu or Fisher models.

In this paper we examine classical economic exchange models in the modern light of social network theory. We are particularly interested in the interaction between the statistical structure of the underlying network and the variation in prices at equilibrium. We quantify the intuition that increased levels of connectivity in the network result in the equalization of

prices, and establish that certain generative models (such as the the *preferential attachment* model of network formation (Barabasi and Albert [1999]) are capable of explaining the heavy-tailed distribution of wealth first observed by Pareto. Closely related work to ours is that of Kranton and Minehart [2001], which also considers networks of buyers and sellers, though they focus more on the economics of network formation.

Many of our results are based on a powerful new *local approximation* method for global equilibrium prices: we show that in the preferential attachment model, prices computed from only local regions of a network yield strikingly good estimates of the global prices. We exploit this method theoretically and computationally. Our study concludes with an application of our model to United Nations international trade data.

## 2   Market Economies on Networks

We first describe the standard *Fisher model*, which consists of a set of *consumers* and a set of *goods*. We assume that there are $g_j$ units of good $j$ in the market, and that each good $j$ is be sold at some price $p_j$. Each consumer $i$ has a cash *endowment* $e_i$, to be used to purchase goods in a manner that maximizes the consumers' utility. In this paper we make the well-studied assumption that the utility function of each consumer is *linear* in the amount of goods consumed (see Gale [1960]), and leave the more general case to future research. Let $u_{ij} \geq 0$ denote the utility derived by $i$ on obtaining a single unit of good $j$. If $i$ consumes $x_{ij}$ amount of good $j$, then the utility $i$ derives is $\sum_j u_{ij}x_{ij}$.

A set of *prices* $\{p_j\}$ and *consumption plans* $\{x_{ij}\}$ constitutes an *equilibrium* if the following two conditions hold:

1. The market *clears*, *i.e.* supply equals demand. More formally, for each $j$, $\sum_i x_{ij} = g_j$.

2. For each consumer $i$, their consumption plan $\{x_{ij}\}_j$ is optimal. By this we mean that the consumption plan maximizes the linear utility function of $i$, subject to the constraint that the total cost of the goods purchased by $i$ is not more than the endowment $e_i$.

It turns out that such an equilibrium always exists if each good $j$ has a consumer which derives nonzero utility for good $j$ — that is, $u_{ij} > 0$ for some $i$ (see Gale [1960]). Furthermore, the equilibrium prices are unique.

We now consider the *graphical Fisher model*, so named because of the introduction of a graph-theoretic or network structure to exchange. In the basic Fisher model, we implicitly assume that all goods are available in a centralized exchange, and all consumers have equal access to these goods. In the graphical Fisher model, we desire to capture the fact that each good may have multiple vendors or *sellers*, and that individual buyers may have access only to some, but not all, of these sellers. There are innumerable settings where such asymmetries arise. Examples include the fact that consumers generally purchase their groceries from local markets, that social connections play a major role in business transactions, and that securities regulations prevent certain pairs of parties from engaging in stock trades.

Without loss of generality, we assume that each seller $j$ sells only one of the available goods. (Each good may have multiple competing sellers.) Let $G$ be a bipartite graph, where buyers and sellers are represented as vertices, and all edges are between a buyer-seller pair. The semantics of the graph are as follows: if there is an edge from buyer $i$ to seller $j$, then buyer $i$ is permitted to purchase from seller $j$. Note that if buyer $i$ is connected to two sellers of the same good, he will always choose to purchase from the cheaper source, since his utility is identical for both sellers (they sell the same good).

The graphical Fisher model is a special case of a more general and recently introduced framework (Kakade et al. [2004]). One of the most interesting features of this model is the fact that at equilibrium, significant price variations can appear solely due to structural properties of the underlying network. We now describe some generative models of economies.

# 3 Generative Models for Social Networks

For simplicity, in the sequel we will consider economies in which the numbers of buyers and sellers are equal. We will also restrict attention to the case in which all sellers sell the *same* good[1].

The simplest generative model for the bipartite graph $G$ might be the *random graph*, in which each edge between a buyer $i$ and a seller $j$ is included independently with probability $p$. This is simply the bipartite version of the classical Erdos-Renyi model (Bollobas [2001]).

Many researchers have sought more realistic models of social network formation, in order to explain observed phenomena such as heavy-tailed degree distributions. We now describe a slight variant of the *preferential attachment* model (see Mitzenmacher [2003]) for the case of a bipartite graph. We start with a graph in which one buyer is connected to one seller. At each *time step*, we add one buyer and one seller as follows. With probability $\alpha$, the buyer is connected to a seller in the existing graph uniformly at random; and with probability $1 - \alpha$, the buyer is connected to a seller chosen *in proportion to the degree* of the seller (preferential attachment). Simultaneously, a seller is attached in a symmetric manner: with probability $\alpha$ the seller is connected to a buyer chosen uniformly at random, and with probability $1 - \alpha$ the seller is connected under preferential attachment. The parameter $\alpha$ in this model thus allows us to move between a pure preferential attachment model ($\alpha = 0$), and a model closer to classical random graph theory ($\alpha = 1$), in which new parties are connected to random extant parties[2].

Note that the above model always produces trees, since the degree of a new party is always 1 upon its introduction to the graph. We thus will also consider a variant of this model in which at each time step, a new seller is still attached to exactly one extant buyer, while each new buyer is connected to $\nu > 1$ extant sellers. The procedure for edge selection is as outlined above, with the modification that the $\nu$ new edges of the buyer are added without replacement — meaning that we resample so that each buyer gets attached to exactly $\nu$ distinct sellers. In a forthcoming long version, we provide results on the statistics of these networks.

The main purpose of the introduction of $\nu$ is to have a model capable of generating highly cyclical (non-tree) networks, while having just a single parameter that can "tune" the asymmetry between the (number of) opportunities for buyers and sellers. There are also economic motivations: it is natural to imagine that new sellers of the good arise only upon obtaining their first customer, but that new buyers arrive already aware of several alternative sellers.

In the sequel, we shall refer to the generative model just described as the *bipartite* $(\alpha, \nu)$-*model*. We will use $n$ to denote the number of buyers and the number of sellers, so the network has $2n$ vertices. Figure 1 and its caption provide an example of a network generated by this model, along with a discussion of its equilibrium properties.

# 4 Economics of the Network: Theory

We now summarize our theoretical findings. The proofs will be provided in a forthcoming long version. We first present a rather intuitive "frontier" theorem, which implies a scheme in which we can find upper and lower bounds on the equilibrium prices using only *local* computations. To state the theorem we require some definitions. First, note that any subset $V'$ of buyers and sellers defines a natural *induced economy*, where the induced graph $G'$

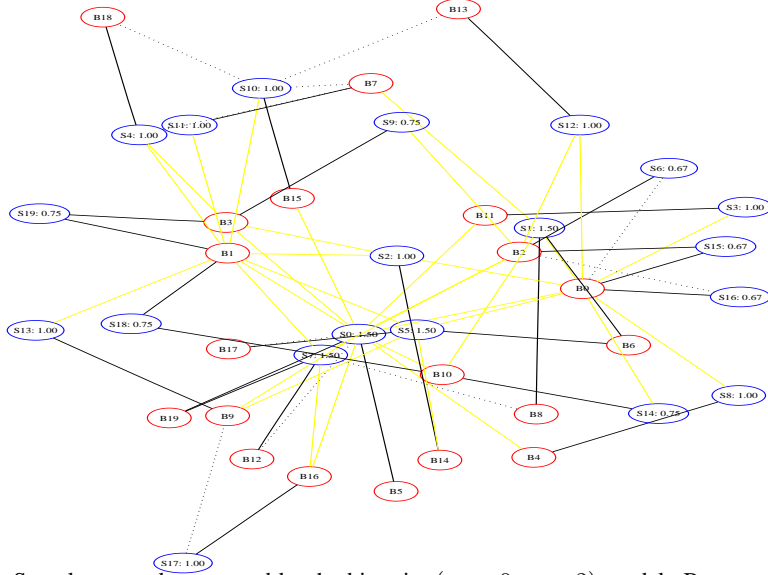

Figure 1: Sample network generated by the bipartite ($\alpha = 0, \nu = 2$)-model. Buyers and sellers are labeled by 'B' or 'S' respectively, followed by an index indicating the time step at which they were introduced to the network. The solid edges in the figure show the *exchange subgraph* —those pairs of buyers and sellers who actually exchange currency and goods at equilibrium. The dotted edges are edges of the network that are unused at equilibrium because they represent inferior prices for the buyers, while the dashed edges are edges of the network that have competitive prices, but are unused at equilibrium due to the specific consumption plan required for market clearance. Each seller is labeled with the price they charge at equilibrium. The example exhibits non-trivial price variation (from 2.00 down to 0.33 per unit good). Note that while there appears to be a correlation between seller degree and price, it is far from a deterministic relation, a topic we shall examine later.

consists of all edges between buyers and sellers in $V'$ that are also in $G$. We say that $G'$ has a *buyer (respectively, seller) frontier* if on every (simple) path in $G$ from a node in $V'$ to a node outside of $V'$, the last node in $V'$ on this path is a buyer (respectively, seller).

**Theorem 1** *(Frontier Bound) If $V'$ has a subgraph $G'$ with a seller (respectively, buyer) frontier, then the equilibrium price of any good $j$ in the induced economy on $V'$ is a lower bound (respectively, upper bound) on the equilibrium price of $j$ in $G$.*

Theorem 1 implies a simple price upper bound: the price commanded by any seller $j$ is bounded by its degree $d$. Although the same upper bound can be seen from first principles, it is instructive to apply Theorem 1. Let $G'$ be the immediate neighborhood of $j$ (which is $j$ and its $d$ buyers); then the equilibrium price in $G'$ is just $d$, since all $d$ buyers are forced to buy from seller $j$. This provides an upper bound since $G'$ has a buyer frontier. Since it can be shown that the degree distribution obeys a power law in the bipartite ($\alpha, \nu$)-model, we have an upper bound on the cumulative price distribution. We use $\beta = (1 - \alpha)\nu/(1 + \nu)$.

**Theorem 2** *In the bipartite $(\alpha, \nu)$-model, the proportion of sellers with price greater than $w$ is $O(w^{-1/\beta})$. For example, if $\alpha = 0$ (pure preferential attachment) and $\nu = 1$, the proportion falls off as $1/w^2$.*

We do not yet have such a closed-form lower bound on the cumulative price distribution. However, as we shall see in Section 5, the price distributions seen in large simulation results do indeed show power-law behavior. Interestingly, this occurs despite the fact that degree is a *poor* predictor of *individual* seller price.

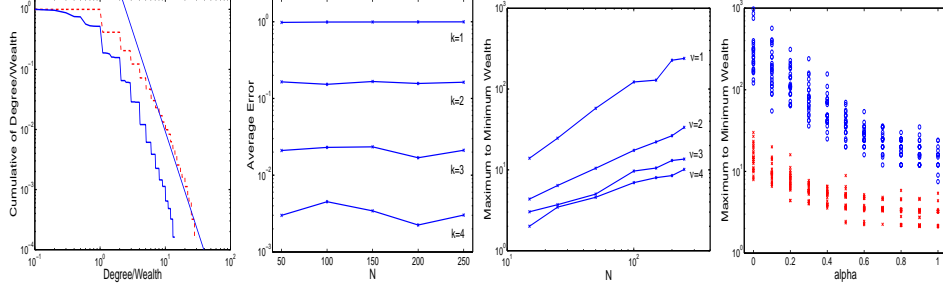

Figure 2: See text for descriptions.

Another quantity of interest is what we might call price variation — the ratio of the price of the richest seller to the poorest seller. The following theorem addresses this.

**Theorem 3** *In the bipartite $(\alpha, \nu)$-model, if $\alpha(\nu^2 + 1) < 1$, then the ratio of the maximum price to the minimum price scales with number of buyers $n$ as $\Omega(n^{\frac{2-\alpha(\nu^2+1)}{1+\nu}})$. For the simplest case in which $\alpha = 0$ and $\nu = 1$, this lower bound is just $\Omega(n)$.*

We conclude our theoretical results with a remark on the price variation in the Erdos-Renyi (random graph) model. First, let us present a condition for there to be no price variation.

**Theorem 4** *A necessary and sufficient condition for there to be* no *price variation, ie for all prices to be equal to* 1*, is that for all sets of vertices $S$, $|N(S)| \geq |S|$, where $N(S)$ is the set of vertices connected by an edge to some vertex in $S$.*

This can be viewed as an extremely weak version of standard *expansion* properties well-studied in graph theory and theoretical computer science — rather than demanding that neighbor sets be strictly larger, we simply ask that they not be smaller. One can further show that for large $n$, the probability that a random graph (for any edge probability $p > 0$) obeys this weak expansion property approaches 1. In other words, in the Erdos-Renyi model, there is no variation in price — a stark contrast to the preferential attachment results.

## 5    Economics of the Network: Simulations

We now present a number of studies on simulated networks (generated according to the bipartite $(\alpha, \nu)$-model). Equilibrium computations were done using the algorithm of Devanur et al. [2002] (or via the application of this algorithm to local subgraphs). We note that it was only the recent development of this algorithm and related ones that made possible the simulations described here (involving hundreds of buyers and sellers in highly cyclical graphs). However, even the speed of this algorithm limits our experiments to networks with $n = 250$ if we wish to run repeated trials to reduce variance. Many of our results suggest that the local approximation schemes discussed below may be far more effective.

**Price and Degree Distributions:**   The first (leftmost) panel of Figure 2 shows empirical *cumulative* price and degree distributions on a loglog scale, averaged over 25 networks drawn according to the bipartite $(\alpha = 0.4, \nu = 1)$-model with $n = 250$. The cumulative degree distribution is shown as a dotted line, where the y-axis represents the fraction of the sellers with degree greater than or equal to $d$, and the degree $d$ is plotted on the x-axis. Similarly, the solid curve plots the fraction of sellers with price greater than some value $w$, where the price $w$ is shown on the x-axis. The thin sold line has our theoretically predicted slope of $\frac{-1}{\beta} = -3.33$, which shows that degree distribution is quite consistent with our expectations, at least in the tails. Though a natural conjecture from the plots is that the price of a seller is essentially determined by its degree, below we will see that the degree

is a rather poor predictor of an individual seller price, while more complex (but still local) properties are extremely accurate predictors.

Perhaps the most interesting finding is that the tail of the *price* distribution looks linear, *i.e.* it also exhibits power law behavior. Our theory provided an upper bound, which is precisely the cumulative degree distribution. We do not yet have a formal lower bound. This plot (and other experiments we have done) further confirm the robustness of the power law behavior in the tail, for $\alpha < 1$ and $\nu = 1$.

As discussed in the Introduction, Pareto's original observation was that the wealth (which corresponds to seller price in our model) distribution in societies obey a power law, which has been born out in many studies on western economies. Since Pareto's original observation, there have been too many explanations of this phenomena to recount here. However, to our knowledge, all of these explanations are more *dynamic* in nature (*eg* a dynamical system of wealth exchange) and don't capture microscopic properties of individual rationality. Here we have power law wealth distribution arising from the combination of certain natural statistical properties of the network, and classical theories of economic equilibrium.

**Bounds via Local Computations:** Recall that Theorem 1 suggests a scheme by which we can do only *local* computations to approximate the *global* equilibrium price for any seller. More precisely, for some seller $j$, consider the subgraph which contains all nodes that are within distance $k$ of $j$. In our bipartite setting, for $k$ odd, this subgraph has a buyer frontier, and for $k$ even, this subgraph has a seller frontier, since we start from a seller. Hence, the equilibrium computation on the odd $k$ (respectively, even $k$) subgraph will provide an upper (respectively, lower) bound.

This provides an heuristic in which one can examine the equilibrium properties of small regions of the graph, without having to do expensive global equilibrium computations. The effectiveness of this heuristic will of course depend on how fast the upper and lower bounds tighten. In general, it is possible to create specific graphs in which these bounds are arbitrarily poor until $k$ is large enough to encompass the entire graph. As we shall see, the performance of this heuristic is dramatically better in the bipartite $(\alpha, \nu)$-model.

The second panel in Figure 2 shows how rapidly the local equilibrium computations converge to the true global equilibrium prices as a function of $k$, and also how this convergence is influenced by $n$. In these experiments, graphs were generated by the bipartite $(\alpha = 0, \nu = 1)$ model. The value of $n$ is given on the x-axis; the average errors (over 5 trials for each value of $k$ and $n$) in the local equilibrium computations are given on the y-axis; and there is a separate plot for each of 4 values for $k$. It appears that for each value of $k$, the quality of approximation obtained has either mild or no dependence on $n$.

Furthermore, the regular spacing of the four plots on the logarithmic scaling of the y-axis establishes the fact that the error of the local approximations is decaying *exponentially* with increased $k$ — indeed, by examining only neighborhoods of 3 steps from a seller in an economy of hundreds, we are already able to compute approximations to global equilibrium prices with errors in the second decimal place. Since the diameter for $n = 250$ was often about 17, this local graph is considerably smaller than the global. However, for the crudest approximation $k = 1$, which corresponds exactly to using seller degree as a proxy for price, we can see that this performs rather poorly. Computationally, we found that the time required to do all 250 local computations for $k = 3$ was about 60% less than the global computation, and would result in presumably greater savings at much larger values of $n$.

**Parameter Dependencies:** We now provide a brief examination of how price variation depends on the parameters of the bipartite $(\alpha, \nu)$-model. We first experimentally evaluate the lower bounds provided in Theorem 3. The third panel of Figure 2 shows the maximum to minimum price as function of $n$ (averaged over 25 trials) on a loglog scale. Each line is for a fixed value of $\nu$, and the values of $\nu$ range form 1 to 4 ($\alpha = 0$).

Recall from Theorem 3, our lower bound on the ratio is $\Omega(n^{\frac{2}{1+\nu}})$ (using $\alpha = 0$). We conjecture that this is tight, and, if so, the slopes of lines (in the loglog plot) should be $\frac{2}{1+\nu}$, which would be $(1, 0.67, 0.5, 0.4)$. The estimated slopes are somewhat close: $(1.02, 0.71, 0.57, 0.53)$. The overall message is that for small values of $\nu$, price variation increases rapidly with the economy size $n$ in preferential attachment.

The rightmost panel of Figure 2 is a scatter plot of $\alpha$ vs. the maximum to minimum price in a graph (where $n = 250$). Here, each point represents the maximum to minimum price ratio in a specific network generated by our model. The circles are for economies generated with $\nu = 1$ and the x's are for economies generated with $\nu = 3$. Here we see that in general, increasing $\alpha$ dramatically decreases price variation (note that the price ratio is plotted on a log scale). This justifies the intuition that as $\alpha$ is increased, more "economic equality" is introduced in the form of less preferential bias in the formation of new edges. Furthermore, the data for $\nu = 1$ shows much larger variation, suggesting that a larger value of $\nu$ also has the effect of equalizing buyer opportunities and therefore prices.

## 6  An Experimental Illustration on International Trade Data

We conclude with a brief experiment exemplifying some of the ideas discussed so far. The statistics division of the United Nations makes available extensive data sets detailing the amounts of trade between major sovereign nations (see http://unstats.un.org/unsd/comtrade). We used a data set indicating, for each pair of nations, the total amount of trade in U.S. dollars between that pair in the year 2002.

For our purposes, we would like to extract a discrete network structure from this numerical data. There are many reasonable ways this could be done; here we describe just one. For each of the 70 largest nations (in terms of total trade), we include connections from that nation to each of its top $k$ trading partners, for some integer $k > 1$. We are thus including the more "important" edges for each nation. Note that each nation will have degree at least $k$, but as we shall see, some nations will have much higher degree, since they frequently occur as a top $k$ partner of other nations. To further cast this extracted network into the bipartite setting we have been considering, we ran many trials in which each nation is randomly assigned a role as either a buyer or seller (which are symmetric roles), and then computed the equilibrium prices of the resulting network economy. We have thus deliberately created an experiment in which *the only economic asymmetries are those determined by the undirected network structure.*

The leftmost panel of Figure 3 show results for 1000 trials under the choice $k = 3$. The upper plot shows the average equilibrium price for each nation, where the nations have been sorted by this average price. We can immediately see that there is dramatic price variation due to the network structure; while many nations suffer equilibrium prices well under \$1, the most topologically favored nations command prices of \$4.42 (U.S.), \$4.01 (Germany), \$3.67 (Italy), \$3.16 (France), \$2.27 (Japan), and \$2.09 (Netherlands). The lower plot of the leftmost panel shows a scatterplot of a nation's degree (x-axis) and its average equilibrium price (y-axis). We see that while there is generally a monotonic relationship, at smaller degree values there can be significant price variation (on the order of \$0.50).

The center panel of Figure 3 shows identical plots for the choice $k = 10$. As suggested by the theory and simulations, increasing the overall connectivity of each party radically reduces price variation, with the highest price being just \$1.10 and the lowest just under \$1. Interestingly, the identities of the nations commanding the highest prices (in order, U.S., France, Switzerland, Germany, Italy, Spain, Netherlands) overlaps significantly with the $k = 3$ case, suggesting a certain robustness in the relative economic status predicted by the model. The lower plot shows that the relationship between degree and price divides the population into "have" (degree above 10) and "have not" (degree below 10) components.

The preponderance of European nations among the top prices suggests our final experi-

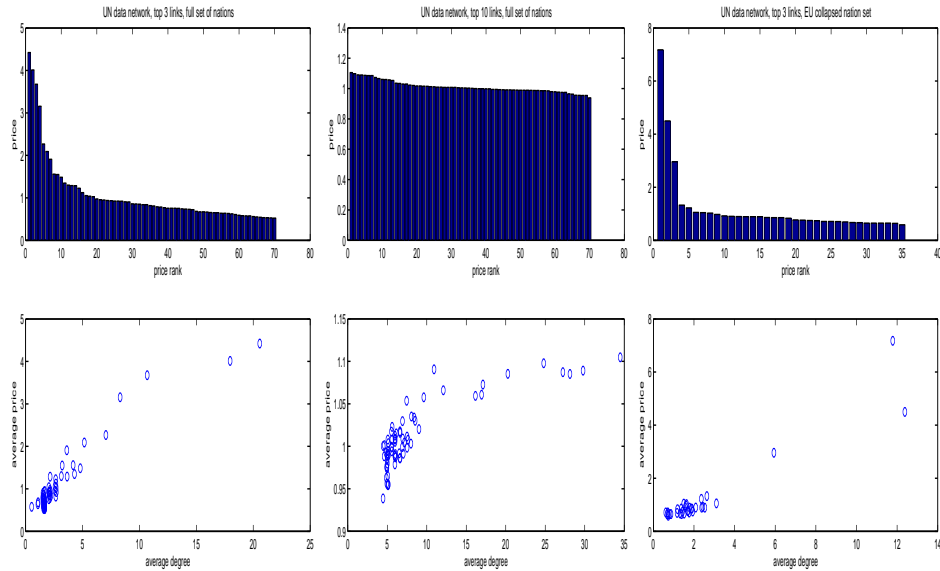

Figure 3: See text for descriptions.

ment, in which we modified the $k = 3$ network by *merging* the 15 current members of the European Union (E.U.) into a single economic nation. This merged vertex has much higher degree than any of its original constituents and can be viewed as an (extremely) idealized experiment in the economic power that might be wielded by a truly unified Europe.

The rightmost panel of Figure 3 provides the results, where we show the relative prices and the degree-price scatterplot for the 35 largest nations. The top prices are now commanded by the E.U. ($7.18), U.S. ($4.50), Japan ($2.96), Turkey ($1.32), and Singapore ($1.22). The scatterplot shows a clear example in which the highest degree (held by the U.S.) does not command the highest price.

### Acknowledgments

We are grateful to Tejas Iyer and Vijay Vazirani for providing their software implementing the Devanur et al. [2002] algorithm. Siddharth Suri acknowledges the support of NIH grant T32HG0046. Robin Pemantle acknowledges the support of NSF grant DMS-0103635.

## Footnotes

[1] From a mathematical and computational standpoint, this restriction is rather weak: when considered in the graphical setting, it already contains the setting of multiple goods with binary utility values, since additional goods can be encoded in the network structure.

[2] We note that $\alpha = 1$ still does not exactly produce the Erdos-Renyi model due to the incremental nature of the network generation: early buyers and sellers are still more likely to have higher degree.

## References

Kenneth J. Arrow and Gerard Debreu. Existence of an equilibrium for a competitive economy. *Econometrica*, 22(3):265–290, July 1954.

A. Barabasi and R. Albert. Emergence of scaling in random networks. *Science*, 286:509–512, 1999.

B. Bollobas. *Random Graphs*. Cambridge University Press, 2001.

Nikhil R. Devanur, Christos H. Papadimitriou, Amin Saberi, and Vijay V. Vazirani. Market equilibrium via a primal-dual-type algorithm. In *FOCS*, 2002.

Irving Fisher. PhD thesis, Yale University, 1891.

D. Gale. *Theory of Linear Economic Models*. McGraw Hill, N.Y., 1960.

Matthew Jackson. A survey of models of network formation: Stability and efficiency. In *Group Formation in Economics: Networks, Clubs and Coalitions*. Cambridge University Press, 2003.

S. Kakade, M. Kearns, and L. Ortiz. Graphical economics. *COLT*, 2004.

R. Kranton and D. Minehart. A theory of buyer-seller networks. *American Economic Review*, 2001.

M. Mitzenmacher. A brief history of generative models for power law and lognormal distributions. *Internet Mathematics*, 1, 2003.